# Learning about Canonical Views from Internet Image Collections

**Elad Mezuman**
Interdisciplinary Center for Neural Computation
Edmond & Lily Safra Center for Brain Sciences
Hebrew University of Jerusalem
http://www.cs.huji.ac.il/~mezuman

**Yair Weiss**
School of Computer Science and Engineering
Edmond & Lily Safra Center for Brain Sciences
Hebrew University of Jerusalem
http://www.cs.huji.ac.il/~yweiss

## Abstract

Although human object recognition is supposedly robust to viewpoint, much research on human perception indicates that there is a preferred or "canonical" view of objects. This phenomenon was discovered more than 30 years ago but the canonical view of only a small number of categories has been validated experimentally. Moreover, the explanation for why humans prefer the canonical view over other views remains elusive. In this paper we ask: Can we use Internet image collections to learn more about canonical views?

We start by manually finding the most common view in the results returned by Internet search engines when queried with the objects used in psychophysical experiments. Our results clearly show that the most likely view in the search engine corresponds to the same view preferred by human subjects in experiments. We also present a simple method to find the most likely view in an image collection and apply it to hundreds of categories. Using the new data we have collected we present strong evidence against the two most prominent formal theories of canonical views and provide novel constraints for new theories.

## 1 Introduction

Images of three dimensional objects exhibit a great deal of variation due to viewpoint. Although ideally object recognition should be viewpoint invariant, much research in human perception indicates that certain views are privileged, or "canonical". As summarized in Blanz et al. [1] there are at least four senses in which a view can be canonical:

- The viewpoint that is assigned the highest goodness rating by participants
- The viewpoint that is first imagined in visual imagery
- The viewpoint that is subjectively selected as the "best" photograph taken with a camera
- The viewpoint found to have the lowest response time and error rate in recognition and naming experiments

The seminal work of Palmer, Rosch and Chase [2] suggested that all of these definitions give the same canonical view. Fig. 1 presents different views of a horse used in their experiments and the average goodness rating given by human subjects. For the horse, the canonical view is a slightly off-axis sideways view, while the least favored view is from above. Subsequent psychological research using slightly different paradigms have mostly supported their conclusions (see [1, 3, 4] for more recent surveys) and expanded it also to scenes rather than just objects [5].

The preference for side views of horses is very robust and can be reliably demonstrated in simple classroom experiments [6]. What makes this view special? Palmer et al. suggested two formal

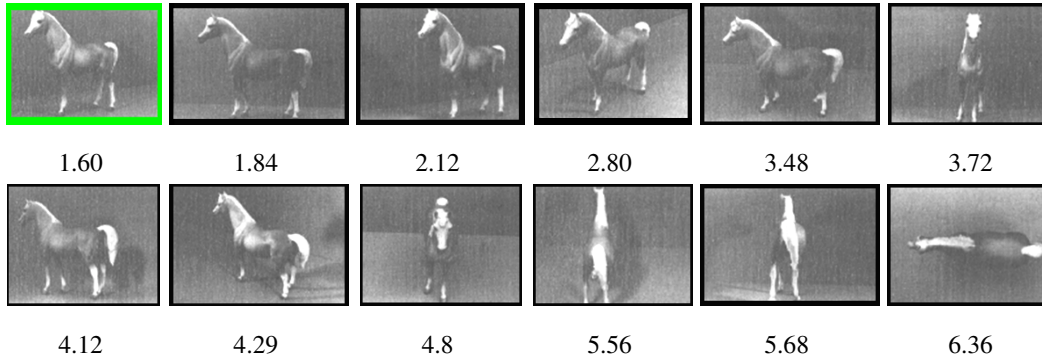

|       |       |       |       |       |       |
|-------|-------|-------|-------|-------|-------|
| 1.60  | 1.84  | 2.12  | 2.80  | 3.48  | 3.72  |
| 4.12  | 4.29  | 4.8   | 5.56  | 5.68  | 6.36  |

Figure 1: When people are asked to rate images of the same object from different views some views consistently get better grades than others. The view that gets the best grade is called the canonical view. The images that were used by Palmer et al. [2] for the horse category in their experiments are presented along with their ratings (1-best, 7-worse).

theories. The first one, called the *frequency hypothesis* argues that the canonical view is the one from which we most often see the object. The second one, called the *maximal information hypothesis* argues that the canonical view is the view that gives the most information about the 3D structure of the object. This view is related to the concept of stable or non-accidental views, i.e. the object will look more or less the same under small transformations of the view. Both of these hypotheses lead to predictions that are testable in principle. If we have access to the statistics with which we view certain objects, we can compute the most frequent view and given the 3D shape of an object we can automatically compute the most stable view [7, 8, 9].

Both of these formal theories have been shown to be insufficient to predict the canonical views preferred by human observers; Palmer et al. [3] presented a small number of counter-examples for each hypothesis. They concluded with the rather vague explanation that: "Canonical views appear to provide the perceiver with what might be called the most diagnostic information about the object: the information that best discriminates it from other objects, derived from the views from which it is most often seen" [3].

One reason for the relative vagueness of theories of canonical views may be the lack of data: the number of objects for which canonical views have been tested in the lab is at most a few dozens. In this paper, we seek to dramatically increase the number of examples for canonical views using Internet search engines and computer vision tools. We expect that since the canonical view of an object corresponds to what people perceive as the "best" photograph, when people include a photograph of an object in their web page, they are most likely to choose a photograph from the canonical view. In other words, we expect the canonical view to be the most frequent view in the set of images retrieved by a search engine when queried for the object.

We start by manually validating our hypothesis and showing that indeed the most frequent view in Internet image collections often corresponds to the cognitive canonical view. We then present an automatic method for finding the most frequent view in a large dataset of images. Rather than trying to map images to views and then finding the most frequent view, we find it by analyzing the density of global image descriptors. Using images for which we have ground truth, we verify that our automatic method indeed finds the most frequent view in a large percentage of the cases. We next apply this method to images retrieved by search engines and find the canonical view for hundreds of categories. Finally we use the canonical views we find to present strong evidence against the two most prominent formal theories of canonical views and provide novel constraints for new theories.

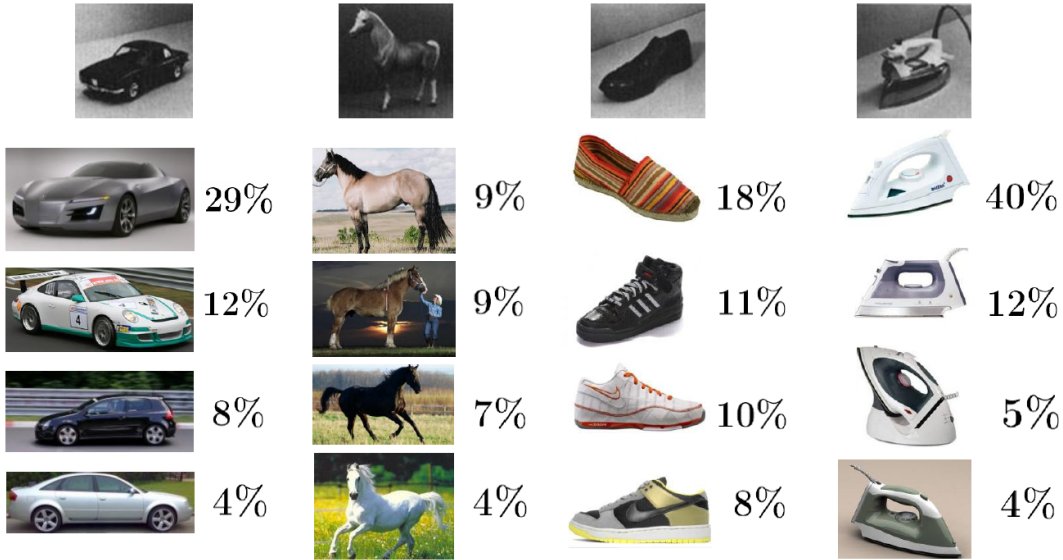

Figure 2: The four most frequent views (frequencies specified) manually found in images returned by Google images (second-fifth rows) often corresponds to the canonical view found in psychophysical experiments (first row).

## 2 Manual experiments with Internet image collections

We first asked whether Internet image collections will show the same view biases as reported in psychophysical experiments. In order to answer this question, we downloaded images of the twelve categories used by Palmer et al. [2] in their psychophysical experiments. To download these images we simply queried Google Image search with the object and retrieved the top returned images.

For each category we manually sorted the images into bins corresponding to similar views (each category could have a different number of bins), counted the number of images in each bin and found the most frequent view. We used 400 images for the four categories presented in Figure 2 and 100 images for the other eight categories. Figure 2 shows the bins with the highest frequencies along with their frequencies and the cognitive canonical view for car, horse, shoe, and steaming iron categories.

The results of this manual experiment are clear cut: for 11 out of the 12 categories, the most frequent view in Google images is the canonical view found by Palmer et al. in the psychophysical experiment (or its mirror view). The only exception is the horse category for which the most frequent view is the one that received the second best ratings in the psychophysical experiments (see figure 1).

This study validates our hypothesis that when humans decide which view of an object to embed in a web page, they exhibit a very similar view bias as is seen in psychophysical experiments. This result now gives us the possibility to harness the huge numbers of images available on the Internet to study these view biases in many categories.

## 3 Can we find the most frequent view automatically?

While the results of the previous section suggests that we can harness Internet image collections, repeating our manual experiment for many categories is impractical. Can we find the most frequent view automatically?

In the computer vision literature we can find several methods to find representative images. Simon et al. [10] showed how clustering Internet photographs of tourist sites can find several "canonical" views of the site. Clustering on images from the Internet is also used to find canonical views (or iconic images) in other works e.g. Berg and Berg [11] and Raguram and Lazebnik [12]. The earlier

work of Denton et al. [13] uses similarity measure between images to find a small subset of canonical images to a larger set of images. The main issue with clustering is that the results depend on the details of the clustering algorithm (initialization, number of clusters etc.) while we look for a method that gives a simple, unique solution. We experimented with clustering methods but found that due to the high variability in our dataset and the difficulty of optimizing the clustering, it was difficult to reliably find clusters that correspond to the most frequent view. Deselaers and Ferrari [14] present a simpler method that finds the image in the center of the GIST image descriptor [15] space to select the prototype image for categories in ImageNet [16]. We experimented with this method and found that often the prototypical image did not correspond to the most frequent view. Jing et al. [17] suggest a method to find a single most representative image (canonical image) for a category relying on similarities between images based on local invariant features. Since they use invariant features the view of the object in the image has no role in the selection of the canonical image. Weyand and Leibe [18] use mode estimation to find iconic images for many images of a single scene using a distance measure based on calculating a homography between the images and measuring the overlap. This is not suitable for our case where we have images of different instances of the same category, not a single rigid scene.

Our method to find the most frequent view is based on estimating the density of views using the Parzen window method, and simply choosing the modes of the density as the most frequent views. If we were given the view of each image as input (e.g. its azimuth and elevation) this would be trivial. In that case the estimated density at point $x$ is $\hat{f}_\sigma(x) = \frac{1}{n}\sum_{i=1}^{n} K_\sigma(x - x_i)$ where $\{x_i\}_{i=1}^{n}$ are the sample points ($x_i$- image $i$, represented using its view) and $K_\sigma(x) = e^{-\|x\|_2^2/2\sigma^2}$.

In real life, of course, the azimuth and elevation are not given as input for each image. One option is to try to compute them. This problem, called pose estimation, is widely studied in computer vision (see [19] for a recent survey for the special case of head poses) and is quite difficult. Here, we take an alternative approach using an attractive feature of the Parzen estimator - it only requires the view similarity between any two images, not the actual views. In other words, if we have an image descriptor so that the distance between descriptors for two images approximates the similarity of views between the objects, we can calculate the Parzen density without ever computing the views.

We chose to use the 512 dimension GIST descriptor [15] which has previously been used to model the similarity between images [12, 14, 20, 21]. The descriptor uses Gabor-like filters on the grayscale image, tuned to 8 orientations at 4 different scales and the average square output on a 4x4 grid for each is its output. This descriptor is pose variant (which is good for our application) but also sensitive to the background (which is bad). We hypothesize that despite this sensitivity to the background, the maximum of the Parzen density when we use GIST similarity between images will serve as a useful proxy for the maximum of the Parzen density when we use view similarity.

### 3.1 Our method

In summary, given an object category our algorithm automatically finds the modes of the GIST distribution in images of that object. However, these modes in the GIST distribution are only approximations to the modes of the view distribution. Our method therefore also includes a manual phase which requires a human to view the output of the algorithm and to verify whether or not this mode in the GIST distribution actually corresponds to a mode in the view distribution.

In the automatic phase we download images for the category (e.g using Google), remove duplicate images and create GIST descriptors for each image. Next we find the two first modes in the GIST space using Parzen window. The first mode is simply the most frequent image in the GIST space and its $k$ closest neighbors. The second mode is the most frequent image that is not a close neighbor of the first most frequent image (e.g not one of its 10% closest neighbors) and its $k$ closest neighbors. For each mode we create a collage of images representing it and this is the output of the first phase (see fig. 4 for example collages). In the second phase a human is required to glance at each collage and to decide if most of the images are from the same view; i.e. a human observer verifies whether the output of the algorithm corresponds to a true point of high density in view space. To validate this second phase, we have conducted several experiments with synthetic images, where the true view distribution is known. We found that when a human verifies that a set of images that are modes in the GIST space are indeed of the same view, then in almost all cases these images are indeed the modes in view space. These experiments are discussed in the supplementary material.

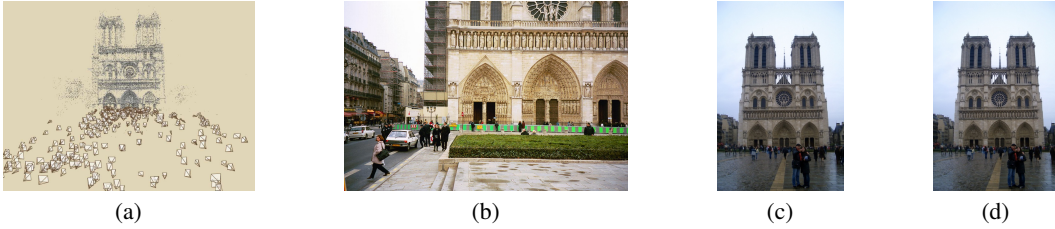

<div align="center">(a)            (b)            (c)            (d)</div>

Figure 3: By using Parzen density estimation on GIST features, we are able to find the most frequent view without calculating the view for a given image. (a) Distribution of views for 715 images of Notre Dame Cathedral in Paris, adapted from [22]. (b) Random image from this dataset. The image from the most frequent view (c) is the same image of the most frequent GIST descriptor (d).

Although the second phase of our method does require human intervention, it requires only a few seconds. This is much less painful than requiring a human to look at all retrieved images which can take a few hours (the automatic part of the method, that finds the modes in GIST space, takes a few seconds of computer time).

## 3.2   Validation

As mentioned above, the main assumption behind our method is that GIST similarity can serve as a proxy for true view similarity. In order to test this assumption, we conducted experiments on datasets where we knew the ground truth distribution of views. In the first experiment, we ran our automatic method on the same images that we manually sorted into views in the previous section: images downloaded from Google image search for the twelve categories used by Palmer et al. in their psychophysical experiments. Results are shown in figure 4. We find that in *10 out of 12 categories* our automatic method found the same most frequent view as we found manually.

In a second experiment, we used the Notre Dame dataset of PhotoTourism [22]. This is a dataset of 715 images of the Notre Dame cathedral taken with consumer cameras. The location of each camera was calculated using bundle adjustment [22]. On this dataset, we calculated the most frequent view using Parzen density estimation in two different ways (1) using the similarity between the camera's rotation matrices and (2) using the GIST similarity between images. As shown in figure 3 *the most frequent view calculated using the two methods was identical*.

## 3.3   Control

As can be seen in figure 4, the most frequent view chosen by our method often has a white, or uniform background. Will a method that simply chooses images with uniform background can also find canonical views? We checked it and this is not the case, among images with smooth backgrounds there is still a large variation in views.

Another possible artifact we considered is the source of the dataset. We wanted to verify we indeed find a global character of the image collections and not a local character of Google. We used our method also on images from ImageNet [16] and Yahoo image search. The ImageNet images were collected by querying various Internet search engines with the desired object, and the resulting set of images was then "cleaned up" by humans. It is important to note that the humans were not instructed to choose particular views but rather to verify that the image contained the desired object. For a subset of the images, ImageNet also supplies bounding boxes around the object of interest; we cropped the objects from the images and considered it as a fourth dataset. There were almost no repeating images between Google, ImageNet and Yahoo datasets. We saw that our method finds preferred views also in the other datasets and that these preferred views are usually the cognitive canonical views. We also saw that using bounding boxes improves the results somewhat. One example of this improvement is the horse category for which we did not find the most frequent view using the the full images but did find it when we used the cropped images.

Results for these control experiment are shown in the supplementary material.

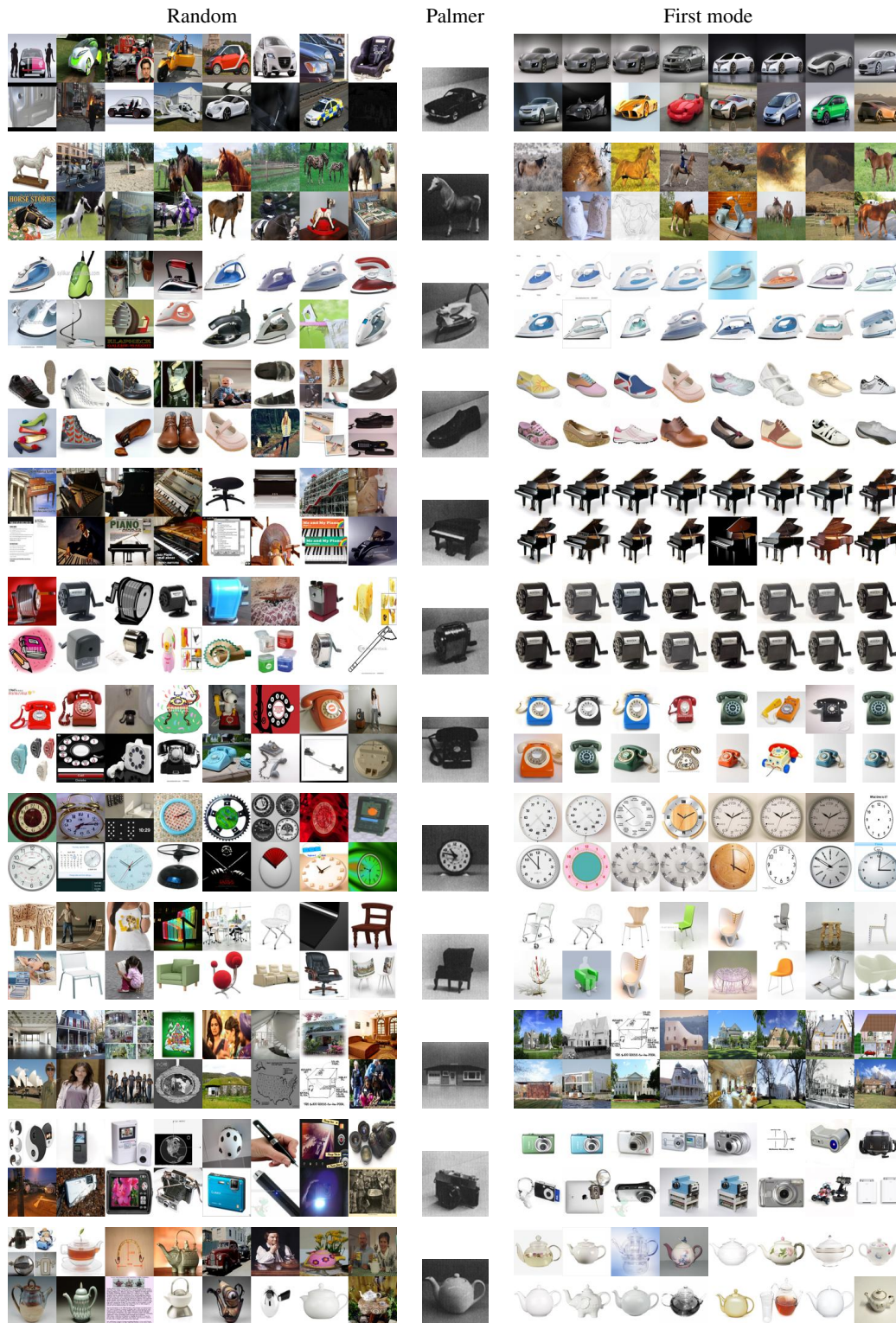

Figure 4: Results on categories we downloaded from Google for which the canonical view was found in Palmer et al. experiments. The collages in the third column are of the first mode of the GIST distribution; the first (top left) image is the most frequent image found where the rest of the images are ordered by their closeness (GIST distance) to the most frequent view

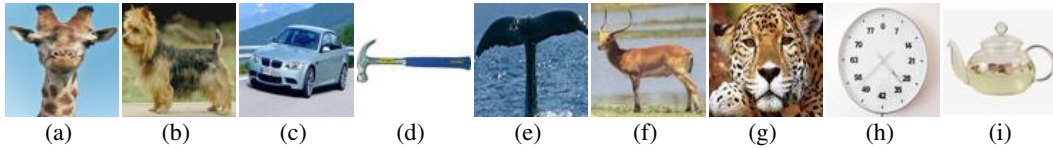

| (a) | (b) | (c) | (d) | (e) | (f) | (g) | (h) | (i) |

Figure 5: Our experiments reveal hundreds of counter-examples against the two most formal theories of canonical views. Prototypical counter-examples found in our experiments for (a-d) the frequency and (e-i) the maximal information hypotheses.

# 4 What can we learn from hundreds of canonical views?

To summarize our validation experiments: although we use GIST similarity as a proxy for view similarity, our method often finds the canonical view. We now turn to use our method on a large number of categories. We used our method to find canonical views for two groups of object categories: (1) 54 categories inspired by the work of Rosch et al. [23], in which human recognition for categories in different levels of abstraction was studied (Rosch's categories). (2) 552 categories of mammals (all the categories of mammals in ImageNet [16] for which there are bounding boxes around the objects), for these categories we used the cropped objects.

For every object category tested we downloaded all corresponding images (in average more than 1,200 images, out of them around 300 with bounding boxes) from ImageNet. The $\sigma$ parameter for the RBF kernel window was fixed for each group of categories and was chosen manually (i.e. we used the same parameter for all the 552 mammal categories but a different one for the Google categories where the data is more noisy). For Rosch's categories we used full images since for some of them bounding boxes are not supplied, for the mammals we used cropped images. For most of the categories the modes found by our algorithm were indeed verified by a human observer as representing a true mode in view space. Thus while our method does not succeed in finding preferred views for all categories, by focusing only on the categories for which humans verified that preferred views were found, we still have canonical views for hundreds of categories. What can we learn from these canonical views?

## 4.1 Do the basic canonical view theories hold?

Palmer et al. [2] raised two basic theories to explain the phenomenon of canonical views: (1) the frequency hypothesis and (2) the maximal information hypothesis. Our experiments reveal hundreds of counter-examples against both theories. We find canonical views of animals that are from the animals' height rather than ours (fig. 5a-b); dogs, for example, are usually seen from above while many of the canonical views we find for dogs are from their height. The canonical views of vehicles are another counter-example for the frequency hypothesis, we usually see vehicles from the side (as pedestrians) or from behind (as drivers), but the canonical views we find are the "perfect" off-axis view (fig. 5a-b). As a third family of examples we have the tools; we usually see them when we use them, this is not the canonical view we find (fig. 5d). For the maximal information hypothesis we find hundreds of counter-examples. While for 20% of the categories we find off-axis canonical views that give the most information about the shape of the object, for more than 60% of the categories we find canonical views that are either side-views (fig. 5f,i) or frontal views (especially views of the face - fig. 5g). Not only do these views not give us the full information about the 3D structure of the object, they are also accidental, i.e. a small change in the view will cause a big change of the appearance of the object; for example in some of the side-views we see only two legs out of four, a small change in the view will reveal the two other legs.

## 4.2 Constraints for new theories

We believe that our experiments reveal several robust features of canonical views that every future theory should take into considerations. The first aspect is that there are several preferred views for a given object. Sometimes these several views are related to symmetry (e.g. a mirror image of the preferred view is also preferred) but in other cases they are different views that are just slightly less preferred than the canonical view (e.g. both the off-axis and the side-view). Another thing we find is that for images of animals, there is a strong preference for photographing just the face (compared to

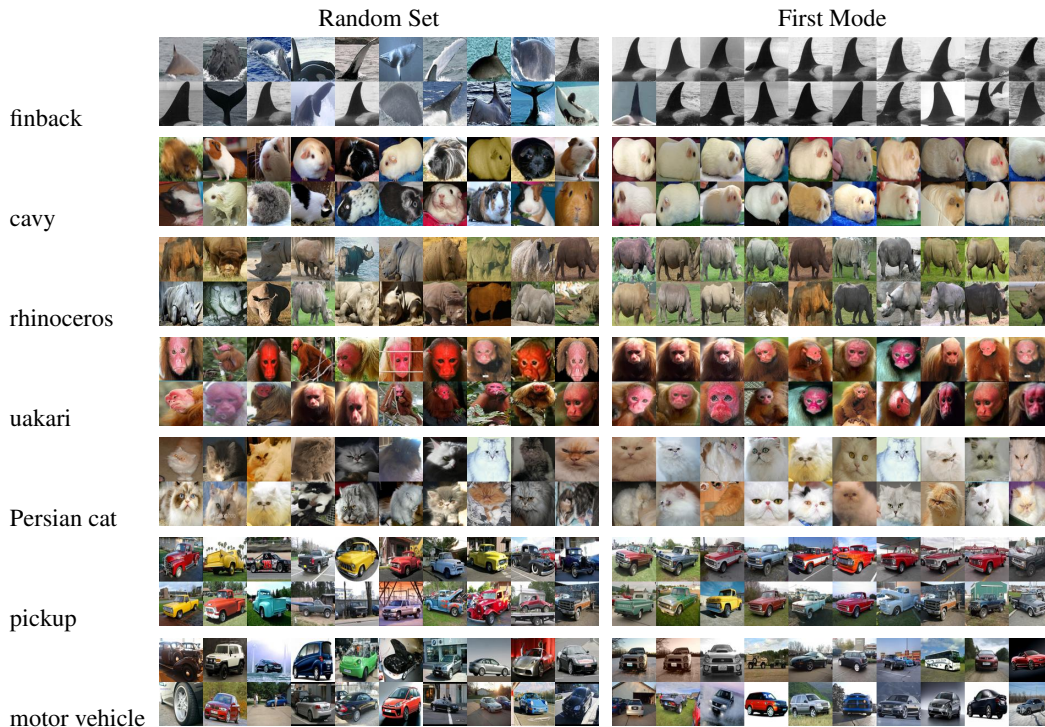

finback

cavy

rhinoceros

uakari

Persian cat

pickup

motor vehicle

Figure 6: Selected collages of the automatic method.

Palmer's result on the horse, where a view just of the face was not given as an option and was hence not preferred). The preference for faces depends on the type of animals (e.g. we find it much more for cats and apes than for big animals like horses). When an animal has very unique features, photographs that include this feature are often preferred. Finally, the view biases are most pronounced for basic and subordinate level categories and less so for superordinate categories (e.g. see motor vehicle in fig. 6). While many of these findings are consistent with the vague theory that "Canonical views appear to provide the perceiver with what might be called the most diagnostic information about the object", we hope that our experimental data with hundreds of categories will enable formalizing these notions into a computational theory.

## 5   Conclusion

In this work we revisited a cognitive phenomenon that was discovered over 30 years ago: a preference by human observers for particular "canonical" views of objects. We showed that a nearly identical view bias can be observed in the results of Internet image search engines, suggesting that when humans decide which image to embed in a web page, they prefer the same canonical view that is assigned highest goodness in laboratory experiments. We presented an automatic method to discover the most likely view in an image collection and used this algorithm to obtain canonical views for hundreds of object categories. Our results provide strong counter-examples for the two formal hypotheses of canonical views; we hope they will serve as a basis for a computational explanation for this fascinating effect.

### Acknowledgments

This work has been supported by the Charitable Gatsby Foundation and the ISF. The authors wish to thank the anonymous reviewers for their helpful comments.

# References

[1] V. Blanz, M.J. Tarr, H.H. Bülthoff, and T. Vetter. What object attributes determine canonical views? *PERCEPTION-LONDON-*, 28:575–600, 1999.

[2] S. Palmer, E. Rosch, and P. Chase. Canonical perspective and the perception of objects. *Attention and performance IX*, pages 135–151, 1981.

[3] S.E. Palmer. *Vision science: Photons to phenomenology*, volume 2. MIT press Cambridge, MA., 1999.

[4] H.H. Bülthoff and S. Edelman. Psychophysical support for a two-dimensional view interpolation theory of object recognition. *Proceedings of the National Academy of Sciences of the United States of America*, 89(1):60, 1992.

[5] K.A. Ehinger and A. Oliva. Canonical views of scenes depend on the shape of the space. *CogSci*, 2011.

[6] A Torralba. Lecture notes on explicit and implicit 3d object models. *http://people.csail.mit.edu/torralba/courses/6.870/slides/lecture4.ppt*.

[7] D. Weinshall and M. Werman. On View Likelihood and Stability. *IEEE Trans. Pattern Anal. Mach. Intell*.

[8] W.T. Freeman. The generic viewpoint assumption in a framework for visual perception. *Nature*, 368(6471).

[9] PM Hall and MJ Owen. Simple canonical views. In *The British Machine Vision Conf.(BMVC05*, volume 1, pages 7–16, 2005.

[10] I. Simon, N. Snavely, and S.M. Seitz. Scene summarization for online image collections. In *Computer Vision, 2007. ICCV 2007. IEEE 11th International Conference on*.

[11] T.L. Berg and A.C. Berg. Finding iconic images. In *CVPR Workshops 2009*.

[12] R. Raguram and S. Lazebnik. Computing iconic summaries of general visual concepts. In *Computer Vision and Pattern Recognition Workshops, 2008. CVPRW'08. IEEE Computer Society Conference on*, pages 1–8. IEEE, 2008.

[13] T. Denton, M.F. Demirci, J. Abrahamson, A. Shokoufandeh, and S. Dickinson. Selecting canonical views for view-based 3-D object recognition. In *ICPR 2004*.

[14] T. Deselaers and V. Ferrari. Visual and semantic similarity in imagenet. In *Computer Vision and Pattern Recognition (CVPR), 2011 IEEE Conference on*, pages 1777–1784. IEEE, 2011.

[15] A. Oliva and A. Torralba. Modeling the shape of the scene: A holistic representation of the spatial envelope. *International Journal of Computer Vision*, 42(3):145–175, 2001.

[16] J. Deng, W. Dong, R. Socher, L.-J. Li, K. Li, and L. Fei-Fei. ImageNet: A Large-Scale Hierarchical Image Database. In *CVPR09*, 2009.

[17] Y. Jing, S. Baluja, and H. Rowley. Canonical image selection from the web. In *Proceedings of the 6th ACM international conference on Image and video retrieval*, pages 280–287. ACM, 2007.

[18] T. Weyand and Leibe. B. Discovering favorite views of popular places with iconoid shift. In *International Conference on Computer Vision (ICCV), 2011 IEEE Conference on*. IEEE, 2011.

[19] E. Murphy-Chutorian and M.M. Trivedi. Head pose estimation in computer vision: A survey. *Pattern Analysis and Machine Intelligence, IEEE Transactions on*, 31(4):607–626, 2009.

[20] M. Douze, H. Jégou, H. Sandhawalia, L. Amsaleg, and C. Schmid. Evaluation of gist descriptors for web-scale image search. In *Proceeding of the ACM International Conference on Image and Video Retrieval*, page 19. ACM, 2009.

[21] J. Xiao, J. Hays, K.A. Ehinger, A. Oliva, and A. Torralba. SUN database: Large-scale scene recognition from abbey to zoo. In *CVPR 2010*.

[22] N. Snavely, S.M. Seitz, and R. Szeliski. Photo tourism: exploring photo collections in 3d. In *ACM Transactions on Graphics (TOG)*, volume 25, pages 835–846. ACM, 2006.

[23] E. Rosch, C.B. Mervis, W.D. Gray, D.M. Johnson, and P. Boyes-Braem. Basic objects in natural categories. *Cognitive psychology*, 8(3):382–439, 1976.

